# Asymptotically Optimal Regularization in Smooth Parametric Models

**Percy Liang**
University of California, Berkeley
pliang@cs.berkeley.edu

**Francis Bach**
INRIA - École Normale Supérieure, France
francis.bach@ens.fr

**Guillaume Bouchard**
Xerox Research Centre Europe, France
Guillaume.Bouchard@xrce.xerox.com

**Michael I. Jordan**
University of California, Berkeley
jordan@cs.berkeley.edu

## Abstract

Many types of regularization schemes have been employed in statistical learning, each motivated by some assumption about the problem domain. In this paper, we present a unified asymptotic analysis of smooth regularizers, which allows us to see how the validity of these assumptions impacts the success of a particular regularizer. In addition, our analysis motivates an algorithm for optimizing regularization parameters, which in turn can be analyzed within our framework. We apply our analysis to several examples, including hybrid generative-discriminative learning and multi-task learning.

## 1   Introduction

Many problems in machine learning and statistics involve the estimation of parameters from finite data. Although empirical risk minimization has favorable limiting properties, it is well known that this procedure can overfit on finite data. Hence, various forms of regularization have been employed to control this overfitting. Regularizers are usually chosen based on assumptions about the problem domain at hand. For example, in classification, we might use $L_2$ regularization if we expect the data to be separable with a large margin. We might regularize with a generative model if we think it is roughly well-specified [7, 20, 15, 17]. In multi-task learning, we might penalize deviation between parameters across tasks if we believe the tasks to be similar [3, 12, 2, 13].

In each case, we would like (1) a procedure for choosing the parameters of the regularizer (for example, its strength) and (2) an analysis that shows the amount by which regularization reduces expected risk, expressed as a function of the compatibility between the regularizer and the problem domain. In this paper, we address these two points by developing an asymptotic analysis of smooth regularizers for parametric problems. The key idea is to derive a second-order Taylor approximation of the expected risk, yielding a simple and interpretable quadratic form which can be directly minimized with respect to the regularization parameters. We first develop the general theory (Section 2) and then apply it to some examples of common regularizers used in practice (Section 3).

## 2   General theory

We use uppercase letters (e.g., $L, R, Z$) to denote random variables and script letters (e.g., $\mathcal{L}, \mathcal{R}, \mathcal{I}$) to denote constant limits of random variables. For a $\lambda$-parametrized differentiable function $\theta \mapsto f(\lambda; \theta)$, let $\dot{f}, \ddot{f}$, and $\dddot{f}$ denote the first, second and third derivatives of $f$ with respect to $\theta$, and let $\nabla f(\lambda; \theta)$ denote the derivative with respect to $\lambda$. Let $X_n = O_p(n^{-\alpha})$ denote a sequence of

random variables for which $n^\alpha X_n$ is bounded in probability. Let $X_n \xrightarrow{P} X$ denote convergence in probability. For a vector $v$, let $v^\otimes = vv^\top$. Expectation and variance operators are denoted as $\mathbb{E}[\cdot]$ and $\mathbb{V}[\cdot]$, respectively.

## 2.1 Setup

We are given a *loss function* $\ell(\cdot; \theta)$ parametrized by $\theta \in \mathbb{R}^d$ (e.g., $\ell((x,y); \theta) = \frac{1}{2}(y - x^\top \theta)^2$ for linear regression). Our goal is to minimize the *expected risk*,

$$\theta_\infty \stackrel{\text{def}}{=} \operatorname*{argmin}_{\theta \in \mathbb{R}^d} \mathcal{L}(\theta), \quad \mathcal{L}(\theta) \stackrel{\text{def}}{=} \mathbb{E}_{Z \sim p^*}[\ell(Z; \theta)], \tag{1}$$

which averages the loss over some true data generating distribution $p^*(Z)$. We do not have access to $p^*$, but instead receive a sample of $n$ i.i.d. data points $Z_1, \ldots, Z_n$ drawn from $p^*$. The standard *unregularized estimator* minimizes the *empirical risk*:

$$\hat{\theta}_n^0 \stackrel{\text{def}}{=} \operatorname*{argmin}_{\theta \in \mathbb{R}^d} L_n(\theta), \quad L_n(\theta) \stackrel{\text{def}}{=} \frac{1}{n} \sum_{i=1}^n \ell(Z_i, \theta). \tag{2}$$

Although $\hat{\theta}_n^0$ is consistent (that is, it converges in probability to $\theta_\infty$) under relatively weak conditions, it is well known that regularization can improve performance substantially for finite $n$. Let $R_n(\lambda, \theta)$ be a (possibly data-dependent) regularization function, where $\lambda \in \mathbb{R}^b$ are the regularization parameters. For linear regression, we might use squared regularization ($R_n(\lambda, \theta) = \frac{\lambda}{2n} \|\theta\|^2$), where $\lambda \in \mathbb{R}$ determines the strength. Define the *regularized estimator* as follows:

$$\hat{\theta}_n^\lambda \stackrel{\text{def}}{=} \operatorname*{argmin}_{\theta \in \mathbb{R}^d} L_n(\theta) + R_n(\lambda, \theta). \tag{3}$$

The goal of this paper is to choose good values of $\lambda$ and analyze the subsequent impact on performance. Specifically, we wish to minimize the *relative risk*:

$$\mathbb{L}_n(\lambda) \stackrel{\text{def}}{=} \mathbb{E}_{Z_1, \ldots, Z_n \sim p^*}[\mathcal{L}(\hat{\theta}_n^\lambda) - \mathcal{L}(\hat{\theta}_n^0)], \tag{4}$$

which is the difference in risk (averaged over the training data) between the regularized and unregularized estimators; $\mathbb{L}_n(\lambda) < 0$ is desirable. Clearly, $\operatorname{argmin}_\lambda \mathbb{L}_n(\lambda)$ is the optimal regularization parameter. However, it is difficult to get a handle on $\mathbb{L}_n(\lambda)$. Therefore, the main focus of this work is on deriving an asymptotic expansion for $\mathbb{L}_n(\lambda)$. In this paper, we make the following assumptions:[1]

**Assumption 1** (Compact support). *The true distribution $p^*(Z)$ has compact support.*

**Assumption 2** (Smooth loss). *The loss function $\ell(z, \theta)$ is thrice-differentiable with respect to $\theta$. Furthermore, assume the expected Hessian of the loss function is positive definite ($\ddot{\mathcal{L}}(\theta_\infty) \succ 0$).[2]*

**Assumption 3** (Smooth regularizer). *The regularizer $R_n(\lambda, \theta)$ is thrice-differentiable with respect to $\theta$ and differentiable with respect to $\lambda$. Assume $R_n(0, \theta) \equiv 0$ and $R_n(\lambda, \theta) \xrightarrow{P} 0$ as $n \to \infty$.*

## 2.2 Rate of regularization strength

Let us establish some basic properties that the regularizer $R_n(\lambda, \theta)$ should satisfy. First, a desirable property is consistency ($\hat{\theta}_n^\lambda \xrightarrow{P} \theta_\infty$), i.e., convergence to the parameters that achieve the minimum possible risk in our hypothesis class. To achieve this, it suffices (and in general also necessitates) that (1) the loss class satisfies standard uniform convergence properties [22] and (2) the regularizer has a vanishing impact in the limit of infinite data ($R_n(\lambda, \theta) \xrightarrow{P} 0$). These two properties can be verified given our assumptions.

The next question is at what rate $R_n(\lambda, \theta)$ should converge to 0? As we show in [16], $R_n(\lambda, \theta) = O_p(n^{-1})$ is the rate that minimizes the relative risk $\mathbb{L}_n$. With this rate, it is natural to consider the regularizer as a prior $p(\theta \mid \lambda) \propto \exp\{-R_n(\lambda, \theta)\}$ (and $-\ell(z, \theta)$ as the log-likelihood), in which case $\hat{\theta}_n^\lambda$ is the maximum a posteriori (MAP) estimate.

## 2.3 Asymptotic expansion

Our main result is the following theorem, which provides a simple interpretable asymptotic expression for the relative risk, characterizing the impact of regularization (see [16] for proof):

**Theorem 1.** *Assume* $R_n(\lambda, \theta_\infty) = O_p(n^{-1})$. *The relative risk admits the following asymptotic expansion:*

$$\mathbb{L}_n(\lambda) = \mathbb{L}(\lambda) \cdot n^{-2} + O_p(n^{-\frac{5}{2}}) \tag{5}$$

*in terms of the asymptotic relative risk:*

$$\mathbb{L}(\lambda) \overset{\text{def}}{=} \frac{1}{2}tr\{\dot{\mathcal{R}}(\lambda)^\otimes \ddot{\mathcal{L}}^{-1}\} - tr\{\mathcal{I}_{\ell\ell}\ddot{\mathcal{L}}^{-1}\ddot{\mathcal{R}}(\lambda)\ddot{\mathcal{L}}^{-1}\} - 2\mathcal{B}^\top\dot{\mathcal{R}}(\lambda) + tr\{\mathcal{I}_{\ell r}(\lambda)\ddot{\mathcal{L}}^{-1}\}, \tag{6}$$

*where* $\ddot{\mathcal{L}} \overset{\text{def}}{=} \mathbb{E}[\ddot{\ell}(Z; \theta_\infty)]$, $\mathcal{R}(\lambda) \overset{\text{def}}{=} \lim_{n\to\infty} nR_n(\lambda, \theta_\infty)$ *(derivatives thereof are defined analogously),* $\mathcal{I}_{\ell\ell} \overset{\text{def}}{=} \mathbb{E}[\dot{\ell}(Z; \theta_\infty)^\otimes]$, $\mathcal{I}_{\ell r}(\lambda) \overset{\text{def}}{=} \lim_{n\to\infty} n\mathbb{E}[\dot{L}_n\dot{R}_n(\lambda)^\top]$, $\mathcal{B} \overset{\text{def}}{=} \lim_{n\to\infty} n\mathbb{E}[\hat{\theta}_n^0 - \theta_\infty]$.

The most important equation of this paper is (6), which captures the lowest-order terms of the relative risk defined in (4).

**Interpretation** The significance of Theorem 1 is in identifying the three problem-dependent contributions to the asymptotic relative risk:

*Squared bias* of the regularizer tr$\{\dot{\mathcal{R}}(\lambda)^\otimes \ddot{\mathcal{L}}^{-1}\}$: $\dot{\mathcal{R}}(\lambda)$ is the gradient of the regularizer at the limiting parameters $\theta_\infty$; the squared regularizer bias is the squared norm of $\dot{\mathcal{R}}(\lambda)$ with respect to the Mahalanobis metric given by $\ddot{\mathcal{L}}$. Note that the squared regularizer bias is always positive: it always increases the risk by an amount which depends on how "wrong" the regularizer is.

*Variance reduction* provided by the regularizer tr$\{\mathcal{I}_{\ell\ell}\ddot{\mathcal{L}}^{-1}\ddot{\mathcal{R}}(\lambda)\ddot{\mathcal{L}}^{-1}\}$: The key quantity is $\ddot{\mathcal{R}}(\lambda)$, the Hessian of the regularizer, whose impact on the relative risk is channeled through $\ddot{\mathcal{L}}^{-1}$ and $\mathcal{I}_{\ell\ell}$. For convex regularizers, $\ddot{\mathcal{R}}(\lambda) \succeq 0$, so we always improve the stability of the estimate by regularizing. Furthermore, if the loss is the negative log-likelihood and our model is *well-specified* (that is, $p^*(z) = \exp\{-\ell(z; \theta_\infty)\}$), then $\mathcal{I}_{\ell\ell} = \ddot{\mathcal{L}}$ by the first Bartlett identity [4], and the variance reduction term simplifies to tr$\{\ddot{\mathcal{R}}(\lambda)\ddot{\mathcal{L}}^{-1}\}$.

*Alignment between regularizer bias and unregularized estimator bias* $2\mathcal{B}^\top\dot{\mathcal{R}}(\lambda) - $ tr$\{\mathcal{I}_{\ell r}(\lambda)\ddot{\mathcal{L}}^{-1}\}$: The alignment has two parts, the first of which is nonzero only for non-linear models and the second of which is nonzero only when the regularizer depends on the training data. The unregularized estimator errs in direction $\mathcal{B}$; we can reduce the risk if the regularizer bias $\dot{\mathcal{R}}(\lambda)$ helps correct for the estimator bias ($\mathcal{B}^\top\dot{\mathcal{R}}(\lambda) > 0$). The second part carries the same intuition: the risk is reduced when the random regularizer compensates for the loss (tr$\{\mathcal{I}_{\ell r}(\lambda)\ddot{\mathcal{L}}^{-1}\} < 0$).

## 2.4 Oracle regularizer

The principal advantage of having a simple expression for $\mathbb{L}(\lambda)$ is that we can minimize it with respect to $\lambda$. Let $\lambda^* \overset{\text{def}}{=} \text{argmin}_\lambda \mathbb{L}(\lambda)$ and call $\hat{\theta}_n^{\lambda^*}$ the *oracle estimator*. We have a closed form for $\lambda^*$ in the important special case that the regularization parameter $\lambda$ is the strength of the regularizer:

**Corollary 1** (Oracle regularization strength). *If* $R_n(\lambda, \theta) = \frac{\lambda}{n}r(\theta)$ *for some* $r(\theta)$, *then*

$$\lambda^* = \underset{\lambda}{argmin}\, \mathbb{L}(\lambda) = \frac{tr\{\mathcal{I}_{\ell\ell}\ddot{\mathcal{L}}^{-1}\ddot{r}\ddot{\mathcal{L}}^{-1}\} + 2\mathcal{B}^\top\dot{r}}{\dot{r}^\top\ddot{\mathcal{L}}^{-1}\dot{r}} \overset{\text{def}}{=} \frac{\mathcal{C}_1}{\mathcal{C}_2}, \quad \mathbb{L}(\lambda^*) = -\frac{\mathcal{C}_1^2}{2\mathcal{C}_2}. \tag{7}$$

*Proof.* (6) is a quadratic in $\lambda$; solve by differentiation. Compute $\mathbb{L}(\lambda^*)$ by substitution. □

In general, $\lambda^*$ will depend on $\theta_\infty$ and hence is not computable from data; Section 2.5 will remedy this. Nevertheless, the oracle regularizer provides an upper bound on performance and some insight into the relevant quantities that make a regularizer useful.

Note $\mathbb{L}(\lambda^*) \leq 0$, since optimizing $\lambda^*$ must be no worse than not regularizing since $\mathbb{L}(0) = 0$. But what might be surprising at first is that the oracle regularization parameter $\lambda^*$ can be negative

| Estimator | UNREGULARIZED | ORACLE | PLUGIN | ORACLEPLUGIN |
|---|---|---|---|---|
| Notation | $\hat{\theta}_n^0$ | $\hat{\theta}_n^{\lambda^*}$ | $\hat{\theta}_n^{\hat{\lambda}_n} = \hat{\theta}_n^{\bullet 1}$ | $\hat{\theta}_n^{\bullet\lambda^{\bullet *}}$ |
| Relative risk | 0 | $\mathbb{L}(\lambda^*)$ | $\mathbb{L}^\bullet(1)$ | $\mathbb{L}^\bullet(\lambda^{\bullet *})$ |

Table 1: Notation for the various estimators and their relative risks.

(corresponding to "anti-regularization"). But if $\frac{\partial \mathbb{L}(\lambda)}{\partial \lambda} = -\mathcal{C}_1 < 0$, then (positive) regularization helps ($\lambda^* > 0$ and $\mathbb{L}(\lambda) < 0$ for $0 < \lambda < 2\lambda^*$).

## 2.5 Plugin regularizer

While the oracle regularizer $R_n(\lambda^*, \theta)$ given by (7) is asymptotically optimal, $\lambda^*$ depends on the unknown $\theta_\infty$, so $\hat{\theta}_n^{\lambda^*}$ is actually not implementable. In this section, we develop the plugin regularizer as a way to avoid this dependence. The key idea is to substitute $\lambda^*$ with an estimate $\hat{\lambda}_n \stackrel{\text{def}}{=} \lambda^* + \varepsilon_n$ where $\varepsilon_n = O_p(n^{-\frac{1}{2}})$. We then use the *plugin estimator* $\hat{\theta}_n^{\hat{\lambda}_n} \stackrel{\text{def}}{=} \operatorname{argmin}_\theta L_n(\theta) + R_n(\hat{\lambda}_n, \theta)$.

How well does this plugin estimator work, that is, what is its relative risk $\mathbb{E}[\mathcal{L}(\hat{\theta}_n^{\hat{\lambda}_n}) - \mathcal{L}(\hat{\theta}_n^0)]$? We cannot simply write $\mathbb{L}_n(\hat{\lambda}_n)$ and apply Theorem 1 because $\mathbb{L}(\cdot)$ can only be applied to non-random arguments. However, we can still leverage existing machinery by defining a new *plugin regularizer* $R_n^\bullet(\lambda^\bullet, \theta) \stackrel{\text{def}}{=} \lambda^\bullet R_n(\hat{\lambda}_n, \theta)$ with regularization parameter $\lambda^\bullet \in \mathbb{R}$. Henceforth, the superscript $\bullet$ will denote quantities concerning the plugin regularizer. The corresponding estimator $\hat{\theta}_n^{\bullet\lambda^\bullet} \stackrel{\text{def}}{=} \operatorname{argmin}_\theta L_n(\theta) + R_n^\bullet(\lambda^\bullet, \theta)$ has relative risk $\mathbb{L}_n^\bullet(\lambda^\bullet) = \mathbb{E}[\mathcal{L}(\hat{\theta}_n^{\bullet\lambda^\bullet}) - \mathcal{L}(\hat{\theta}_n^{\bullet 0})]$. The key identity is $\hat{\theta}_n^{\hat{\lambda}_n} = \hat{\theta}_n^{\bullet 1}$, which means the asymptotic risk of the plugin estimator $\hat{\theta}_n^{\hat{\lambda}_n}$ is simply $\mathbb{L}^\bullet(1)$.

We could try to squeeze more out of the plugin regularizer by further optimizing $\lambda^\bullet$ according to $\lambda^{\bullet *} \stackrel{\text{def}}{=} \operatorname{argmin}_{\lambda^\bullet} \mathbb{L}^\bullet(\lambda^\bullet)$ and use the *oracle plugin estimator* $\hat{\theta}_n^{\bullet\lambda^{\bullet *}}$ rather than just using $\lambda^\bullet = 1$. In general, this is not useful since $\lambda^{\bullet *}$ might depend on $\theta_\infty$, and the whole point of plugin is to remove this dependence. However, in a fortuitous turn of events, for some linear models (Sections 3.1 and 3.4), $\lambda^{\bullet *}$ is in fact independent of $\theta_\infty$, and so $\hat{\theta}_n^{\bullet\lambda^{\bullet *}}$ is actually implementable. Table 1 summarizes all the estimators we have discussed.

The following theorem relates the risks of all estimators we have considered (see [16] for the proof):

**Theorem 2** (Relative risk of plugin). *The relative risk of the plugin estimator is $\mathbb{L}^\bullet(1) = \mathbb{L}(\lambda^*) + \mathcal{E}$, where $\mathcal{E} \stackrel{\text{def}}{=} \lim_{n\to\infty} n\mathbb{E}[tr\{\dot{L}_n(\nabla\dot{R}_n(\lambda^*)\varepsilon_n)^\top \ddot{\mathcal{L}}^{-1}\}]$. If $R_n(\lambda)$ is linear in $\lambda$, then the relative risk of the oracle plugin estimator is $\mathbb{L}^\bullet(\lambda^{\bullet *}) = \mathbb{L}^\bullet(1) + \frac{\mathcal{E}^2}{4\mathbb{L}(\lambda^*)}$ with $\lambda^{\bullet *} = 1 + \frac{\mathcal{E}}{2\mathbb{L}(\lambda^*)}$.*

Note that the sign of $\mathcal{E}$ depends on the nature of the error $\varepsilon_n$, so PLUGIN could be either better or worse than ORACLE. On the other hand, ORACLEPLUGIN is always better than PLUGIN. We can get a simpler expression for $\mathcal{E}$ if we know more about $\varepsilon_n$ (see [16] for the proof):

**Theorem 3.** *Suppose $\lambda^* = f(\theta_\infty)$ for some differentiable $f : \mathbb{R}^d \to \mathbb{R}^b$. If $\hat{\lambda}_n = f(\hat{\theta}_n^0)$, then the results of Theorem 2 hold with $\mathcal{E} = -tr\{\mathcal{I}_{\ell\ell}\ddot{\mathcal{L}}^{-1}\nabla\dot{\mathcal{R}}(\lambda^*)\dot{f}\ddot{\mathcal{L}}^{-1}\}$.*

## 3 Examples

In this section, we apply our results from Section 2 to specific problems. Having made all the asymptotic derivations in the general setting, we now only need to make a few straightforward calculations to obtain the asymptotic relative risks and regularization parameters for a given problem. We first explore two classical examples from statistics (Sections 3.1 and 3.2) to get some intuition for the theory. Then we consider two important examples in machine learning (Sections 3.3 and 3.4).

### 3.1 Estimation of normal means

Assume that data are generated from a multivariate normal distribution with $d$ independent components ($p^* = \mathcal{N}(\theta_\infty, I)$). We use the negative log-likelihood as the loss function: $\ell(x; \theta) = \frac{1}{2}(x-\theta)^2$, so the model is well-specified.

In his seminal 1961 paper [14], Stein showed that, surprisingly, the standard empirical risk minimizer $\hat{\theta}_n^0 = \bar{X} \stackrel{\text{def}}{=} \frac{1}{n}\sum_{i=1}^n X_i$ is beaten by the James-Stein estimator $\hat{\theta}_n^{\text{JS}} \stackrel{\text{def}}{=} \bar{X}\left(1 - \frac{d-2}{n\|\bar{X}\|^2}\right)$ in the sense that $\mathbb{E}[\mathcal{L}(\hat{\theta}_n^{\text{JS}})] < \mathbb{E}[\mathcal{L}(\hat{\theta}_n^0)]$ for all $n$ and $\theta_\infty$ if $d > 2$. We will show that the James-Stein estimator is essentially equivalent to ORACLEPLUGIN with quadratic regularization ($r(\theta) = \frac{1}{2}\|\theta\|^2$).

First compute $\dot{L}_n = \theta_\infty - \bar{X}$, $\ddot{\mathcal{L}} = I$, $\mathcal{B} = 0$, $\dot{r} = \theta_\infty$, and $\ddot{r} = I$. By (7), the oracle regularization weight is $\lambda^* = \frac{d}{\|\theta_\infty\|^2}$, which yields a relative risk of $\mathbb{L}(\lambda^*) = -\frac{d^2}{2\|\theta_\infty\|^2}$.

Now let us derive PLUGIN (Section 2.5). We have $f(\theta) = \frac{d}{\|\theta\|^2}$ and $\dot{f}(\theta) = \frac{-2d\theta}{\|\theta\|^4}$. By Theorems 2 and 3, $\mathcal{E} = \frac{2d}{\|\theta_\infty\|^2}$ and $\mathbb{L}^\bullet(1) = -\frac{d(d-4)}{2\|\theta_\infty\|^2}$. Note that since $\mathcal{E} > 0$, PLUGIN is always (asymptotically) worse than ORACLE but better than UNREGULARIZED if $d > 4$.

To get ORACLEPLUGIN, compute $\lambda^{\bullet*} = 1 - \frac{2}{d}$ (note that this doesn't depend on $\theta_\infty$), which results in $R_n^\bullet(\theta) = \frac{1}{2}\frac{1-\frac{2}{d}}{\|\bar{X}\|^2}\|\theta\|^2$. By Theorem 2, its relative risk is $\mathbb{L}^\bullet(\lambda^{\bullet*}) = -\frac{(d-2)^2}{2\|\theta_\infty\|^2}$, which offers a small improvement over PLUGIN (and is superior to UNREGULARIZED when $d > 2$).

Note that the ORACLEPLUGIN and PLUGIN are adaptive: We regularize more or less depending on whether our preliminary estimate of $\bar{X}$ is small or large, respectively. By simple algebra, ORACLEPLUGIN has a closed form $\hat{\theta}_n^{\bullet\lambda^{\bullet*}} = \bar{X}\left(1 - \frac{d-2}{n\|\bar{X}\|^2+d-2}\right)$, which differs from JAMESSTEIN by a very small amount: $\hat{\theta}_n^{\bullet\lambda^{\bullet*}} - \hat{\theta}_n^{\text{JS}} = O_p(n^{-\frac{5}{2}})$. ORACLEPLUGIN has the added benefit that it always shrinks towards zero by an amount between 0 and 1, whereas JAMESSTEIN can overshoot. Empirically, we found that ORACLEPLUGIN generally had a lower expected risk than JAMESSTEIN when $\|\theta_\infty\|$ is large, but JAMESSTEIN was better when $\|\theta_\infty\| \leq 1$.

## 3.2 Binomial estimation

Consider the estimation of $\theta$, the log-odds of a coin coming up heads. We use the negative log-likelihood loss $\ell(x; \theta) = -x\theta + \log(1 + e^\theta)$, where $x \in \{0, 1\}$ is the outcome of the coin. This example serves to provide intuition for the bias $\mathcal{B}$ appearing in (6), which is typically ignored in first-order asymptotics or is zero (for linear models).

Consider a regularizer $r(\theta) = \frac{1}{2}(\theta + 2\log(1 + e^{-\theta}))$, which corresponds to a Beta$(\frac{\lambda}{2}, \frac{\lambda}{2})$ prior. Choosing $\lambda$ has been studied extensively in statistics. Some common choices are the Haldane prior ($\lambda = 0$), the reference (Jeffreys) prior ($\lambda = 1$), the uniform prior ($\lambda = 2$), and Laplace smoothing ($\lambda = 4$). We will choose $\lambda$ to minimize expected risk adaptively based on data.

Define $\mu \stackrel{\text{def}}{=} \frac{1}{1+e^{-\theta_\infty}}$, $v \stackrel{\text{def}}{=} \mu(1-\mu)$, and $b \stackrel{\text{def}}{=} \mu - \frac{1}{2}$. Then compute $\ddot{\mathcal{L}} = v$, $\dddot{\mathcal{L}} = -2vb$, $\dot{r} = b$, $\ddot{r} = v$, $\mathcal{B} = -v^{-1}b$. ORACLE corresponds to $\lambda^* = 2 + \frac{v}{b^2}$. Note that $\lambda^* > 0$, so again (positive) regularization always helps.

We can compute the difference between ORACLE and PLUGIN: $\mathcal{E} = 2 - \frac{2v}{b^2}$. If $|b| > \frac{\sqrt{2}}{4}$, $\mathcal{E} > 0$, which means that PLUGIN is worse; otherwise PLUGIN is actually better. Even when PLUGIN is worse than ORACLE, PLUGIN is still better than UNREGULARIZED, which can be verified by checking that $\mathbb{L}^\bullet(1) = -\frac{5}{2}vb^{-2} - 2v^{-1}b^2 < 0$ for all $\theta_\infty$.

## 3.3 Hybrid generative-discriminative learning

In prediction tasks, we wish to learn a mapping from some input $x \in \mathcal{X}$ to an output $y \in \mathcal{Y}$. A common approach is to use probabilistic models defined by exponential families, which is defined by a vector of sufficient statistics (features) $\phi(x, y) \in \mathbb{R}^d$ and an accompanying vector of parameters $\theta \in \mathbb{R}^d$. These features can be used to define a generative model (8) or a discriminative model (9):

$$p_\theta(x, y) = \exp\{\phi(x, y)^\top \theta - A(\theta)\}, \quad A(\theta) = \log \int_{\mathcal{X}} \int_{\mathcal{Y}} \exp\{\phi(x, y)^\top \theta\} dy dx, \quad (8)$$

$$p_\theta(y \mid x) = \exp\{\phi(x, y)^\top \theta - A(\theta; x)\}, \quad A(\theta; x) = \log \int_{\mathcal{Y}} \exp\{\phi(x, y)^\top \theta\} dy. \quad (9)$$

| Misspecification | $\mathrm{tr}\{\mathcal{I}_{\ell\ell}v_x^{-1}vv_x^{-1}\}$ | $2\mathcal{B}^\top(\mu-\mu_{xy})$ | $\mathrm{tr}\{(\mu-\mu_{xy})^{\otimes}v_x^{-1}\}$ | $\lambda^*$ | $\mathbb{L}(\lambda^*)$ |
|---|---|---|---|---|---|
| 0% | 5 | 0 | 0 | $\infty$ | -0.65 |
| 5% | 5.38 | -0.073 | 0.00098 | 310 | -48 |
| 50% | 13.8 | -1.0 | 0.034 | 230 | -808 |

Table 2: The oracle regularizer for the hybrid generative-discriminative estimator. As misspecification increases, we regularize less, but the relative risk is reduced more (due to more variance reduction).

Given these definitions, we can either use a generative estimator $\hat{\theta}_n^{\mathrm{gen}} \overset{\mathrm{def}}{=} \mathrm{argmin}_\theta\, G_n(\theta)$, where $G_n(\theta) = -\frac{1}{n}\sum_{i=1}^n \log p_\theta(x,y)$ or a discriminative estimator $\hat{\theta}_n^{\mathrm{dis}} \overset{\mathrm{def}}{=} \mathrm{argmin}_\theta\, D_n(\theta)$, where $D_n(\theta) = -\frac{1}{n}\sum_{i=1}^n \log p_\theta(y \mid x)$.

There has been a flurry of work on combining generative and discriminative learning [7, 20, 15, 18, 17]. [17] showed that if the generative model is well-specified ($p^*(x,y) = p_{\theta_\infty}(x,y)$), then the generative estimator is better in the sense that $\mathcal{L}(\hat{\theta}_n^{\mathrm{gen}}) \leq \mathcal{L}(\hat{\theta}_n^{\mathrm{dis}}) - \frac{c}{n} + O_p(n^{-\frac{3}{2}})$ for some $c \geq 0$; if the model is misspecified, the discriminative estimator is asymptotically better. To create a hybrid estimator, let us treat the discriminative and generative objectives as the empirical risk and the regularizer, respectively, so $\ell((x,y);\theta) = -\log p_\theta(y \mid x)$, so $L_n = D_n$ and $R_n(\lambda,\theta) = \frac{\lambda}{n}G_n(\theta)$. As $n \to \infty$, the discriminative objective dominates as desired. Our approach generalizes the analysis of [6], which applies only to unbiased estimators for conditionally well-specified models.

By moment-generating properties of the exponential family, we arrive at the following quantities (write $\phi$ for $\phi(X,Y)$): $\ddot{\mathcal{L}} = v_x \overset{\mathrm{def}}{=} \mathbb{E}_{p^*(X)}[\mathbb{V}_{p_{\theta_\infty}(Y|X)}[\phi|X]]$, $\dot{\mathcal{R}}(\lambda) = \lambda(\mu-\mu_{xy}) \overset{\mathrm{def}}{=} \lambda(\mathbb{E}_{p_{\theta_\infty}(X,Y)}[\phi] - \mathbb{E}_{p^*(X,Y)}[\phi])$, and $\ddot{\mathcal{R}}(\lambda) = \lambda v \overset{\mathrm{def}}{=} \lambda\mathbb{V}_{p_{\theta_\infty}(X,Y)}[\phi]$. The oracle regularization parameter is then

$$\lambda^* = \frac{\mathrm{tr}\{\mathcal{I}_{\ell\ell}v_x^{-1}vv_x^{-1}\} + 2\mathcal{B}^\top(\mu-\mu_{xy}) - \mathrm{tr}\{\mathcal{I}_{\ell r}v_x^{-1}\}}{\mathrm{tr}\{(\mu-\mu_{xy})^{\otimes}v_x^{-1}\}}. \tag{10}$$

The sign and magnitude of $\lambda^*$ provides insight into how generative regularization improves prediction as a function of the model and problem: Specifically, a large positive $\lambda^*$ suggests regularization is helpful. To simplify, assume that the discriminative model is well-specified, that is, $p^*(y \mid x) = p_{\theta_\infty}(y \mid x)$ (note that the generative model could still be misspecified). In this case, $\mathcal{I}_{\ell\ell} = \ddot{\mathcal{L}}$, $\mathcal{I}_{\ell r} = v_x$, and so the numerator reduces to $\mathrm{tr}\{(v - v_x)v_x^{-1}\} + 2\mathcal{B}^\top(\mu-\mu_{xy})$.

Since $v \succeq v_x$ (the key fact used in [17]), the variance reduction (plus the random alignment term from $\mathcal{I}_{\ell r}$) is always non-negative with magnitude equal to the fraction of missing information provided by the generative model. There is still the non-random alignment term $2\mathcal{B}^\top(\mu-\mu_{xy})$, whose sign depends on the problem. Finally, the denominator (always positive) affects the optimal magnitude of the regularization. If the generative model is almost well-specified, $\mu$ will be close to $\mu_{xy}$, and the regularizer should be trusted more (large $\lambda^*$). Since our analysis is local, misspecification (how much $p_{\theta_\infty}(x,y)$ deviates from $p^*(x,y)$) is measured by a Mahalanobis distance between $\mu$ and $\mu_{xy}$, rather than something more stringent and global like KL-divergence.

**An empirical example**   To provide some concrete intuition, we investigated the oracle regularizer for a synthetic binary classification problem of predicting $y \in \{0,1\}$ from $x \in \{0,1\}^k$. Using features $\phi(x,y) = (\mathbb{I}[y=0]x^\top, \mathbb{I}[y=1]x^\top)^\top$ defines the corresponding generative (Naive Bayes) and discriminative (logistic regression) estimators. We set $k = 5$, $\theta_\infty = (\frac{1}{10}, \cdots, \frac{1}{10}, \frac{3}{10}, \cdots, \frac{3}{10})^\top$, and $p^*(x,y) = (1-\varepsilon)p_{\theta_\infty}(x,y) + \varepsilon p_{\theta_\infty}(y)p_{\theta_\infty}(x_1 \mid y)\mathbb{I}[x_1 = \cdots = x_k]$. The amount of misspecification is controlled by $0 \leq \varepsilon \leq 1$, the fraction of examples whose features are perfectly correlated.

Table 2 shows how the oracle regularizer changes with $\varepsilon$. As $\varepsilon$ increases, $\lambda^*$ decreases (we regularize less) as expected. But perhaps surprisingly, the relative risk is reduced with more misspecification; this is due to the fact that the variance reduction term increases and has a quadratic effect on $\mathbb{L}(\lambda^*)$.

Figure 1(a) shows the relative risk $\mathbb{L}_n(\lambda)$ for various values of $\lambda$. The vertical line corresponds to $\lambda^*$, which was computed numerically by sampling. Note that the minimum of the curves

($\text{argmin}_\lambda \mathbb{L}_n(\lambda)$), the desired quantity, is quite close to $\lambda^*$ and approaches $\lambda^*$ as $n$ increases, which empirically justifies our asymptotic approximations.

**Unlabeled data**  One of the main advantages of having a generative model is that we can leverage unlabeled examples by marginalizing out their hidden outputs. Specifically, suppose we have $m$ i.i.d. unlabeled examples $X_{n+1}, \ldots, X_{n+m} \sim p^*(x)$, with $m \to \infty$ as $n \to \infty$. Define the unlabeled regularizer as $R_n(\lambda, \theta) = -\frac{\lambda}{nm} \sum_{i=1}^m \log p_\theta(X_{n+i})$.

We can compute $\dot{\mathcal{R}} = \mu - \mu_{xy}$ using the stationary conditions of the loss function at $\theta_\infty$. Also, $\ddot{\mathcal{R}} = v - v_x$, and $\mathcal{I}_{\ell r} = 0$ (the regularizer doesn't depend on the labeled data). If the model is conditionally well-specified, we can verify that the oracle regularization parameter $\lambda^*$ is the same as if we had regularized with $G_n$. This equivalence suggests that the dominant concern asymptotically is developing an adequate generative model with small bias and not exactly how it is used in learning.

## 3.4  Multi-task regression

The intuition behind multi-task learning is to share statistical strength between tasks [3, 12, 2, 13]. Suppose we have $K$ regression tasks. For each task $k = 1, \ldots, K$, we generate each data point $i = 1, \ldots, n$ independently as follows: $X_i^k \sim p^*(X_i^k)$ and $Y_i^k \sim \mathcal{N}(X_i^{k\top}\theta_\infty^k, 1)$. We can treat this as a single task problem by concatenating the vectors for all the tasks: $X_i = (X_i^{1\top}, \ldots, X_i^{K\top})^\top \in \mathbb{R}^{Kd}$, $Y = (Y^1, \ldots, Y^K)^\top \in \mathbb{R}^K$, and $\theta = (\theta^{1\top}, \ldots, \theta^{K\top})^\top \in \mathbb{R}^{Kd}$. It will also be useful to represent $\theta \in \mathbb{R}^{Kd}$ by the matrix $\Theta = (\theta^1, \ldots, \theta^K) \in \mathbb{R}^{d \times K}$. The loss function is $\ell((x,y), \theta) = \frac{1}{2} \sum_{k=1}^K (y^k - x^{k\top}\theta^k)^2$. Assume the model is conditionally well-specified.

We would like to be flexible in case some tasks are more related than others, so let us define a positive definite matrix $\Lambda \in \mathbb{R}^{K \times K}$ of inter-task affinities and use the quadratic regularizer: $r(\Lambda, \theta) = \frac{1}{2}\theta^\top (\Lambda \otimes I_d)\theta$. For simplicity, assume $\mathbb{E}X_i^{k\otimes} = I_d$, which implies that $\mathcal{I}_{\ell\ell} = \ddot{\mathcal{L}} = I_{Kd}$.

Most of the computations that follow parallel those of Section 3.1, only extended to matrices. Substituting the relevant quantities into (6) yields the relative risk: $\mathbb{L}(\Lambda) = \frac{1}{2}\text{tr}\{\Lambda^2 \Theta_\infty^\top \Theta_\infty\} - d\text{tr}\{\Lambda\}$. Optimizing with respect to $\Lambda$ produces the oracle regularization parameter $\Lambda^* = d(\Theta_\infty^\top \Theta_\infty)^{-1}$ and its associated relative risk $\mathbb{L}(\Lambda^*) = -\frac{1}{2}d^2\text{tr}\{(\Theta_\infty^\top \Theta_\infty)^{-1}\}$.

To analyze PLUGIN, first compute $\dot{f} = -d(\Theta_\infty^\top \Theta_\infty)^{-1}(2\Theta_\infty^\top(\cdot))(\Theta_\infty^\top \Theta_\infty)^{-1}$; we find that PLUGIN increases the asymptotic risk by $\mathcal{E} = 2d\text{tr}\{(\Theta_\infty^\top \Theta_\infty)^{-1}\}$. However, the relative risk of PLUGIN is still favorable when $d > 4$, as $\mathbb{L}^\bullet(1) = -\frac{1}{2}d(d-4)\text{tr}\{(\Theta_\infty^\top \Theta_\infty)^{-1}\} < 0$ for $d > 4$.

We can do slightly better using ORACLEPLUGIN ($\lambda^{\bullet*} = 1 - \frac{2}{d}$), which results in a relative risk of $\mathbb{L}^\bullet(\lambda^{\bullet*}) = -\frac{1}{2}(d-2)^2\text{tr}\{(\Theta_\infty^\top \Theta_\infty)^{-1}\}$. For comparison, if we had solved the $K$ regression tasks completely independently with $K$ independent regularization parameters, our relative risk would have been $-\frac{1}{2}(d-2)^2(\sum_{k=1}^K \|\theta_\infty^k\|^{-2})$ (following similar but simpler computations).

We now compare joint versus independent regularization. Let $A = \Theta_\infty^\top \Theta_\infty$ with eigendecomposition $A = UDU^\top$. The difference in relative risks between joint and independent regularization is $\Delta = -\frac{1}{2}(d-2)^2(\sum_k D_{kk}^{-1} - \sum_k A_{kk}^{-1})$ ($\Delta < 0$ means joint regularization is better). The gap between joint and independent regularization is large when the tasks are non-trivial but similar ($\theta_\infty^k$s are close, but $\|\theta_\infty^k\|$ is large). In that case, $D_{kk}^{-1}$ is quite large for $k > 1$, but all the $A_{kk}^{-1}$s are small.

**MHC-I binding prediction**  We evaluated our multitask regularization method on the IEDB MHC-I peptide binding dataset created by [19] and used by [13]. The goal here is to predict the binding affinity (represented by $\log \text{IC}_{50}$) of a MHC-I molecule given its amino-acid sequence (represented by a vector of binary features, reduced to a 20-dimensional real vector using SVD). We created five regression tasks corresponding to the five most common MHC-I molecules.

We compared four estimators: UNREGULARIZED, DIAGCV ($\Lambda = cI$), UNIFORMCV (using the same task-affinity for all pairs of tasks with $\Lambda = c(\mathbf{1}^\otimes + 10^{-5}I)$), and PLUGINCV ($\Lambda = cd(\hat{\Theta}_n^\top \hat{\Theta}_n)^{-1}$), where $c$ was chosen by cross-validation.[3] Figure 1 shows the results averaged over

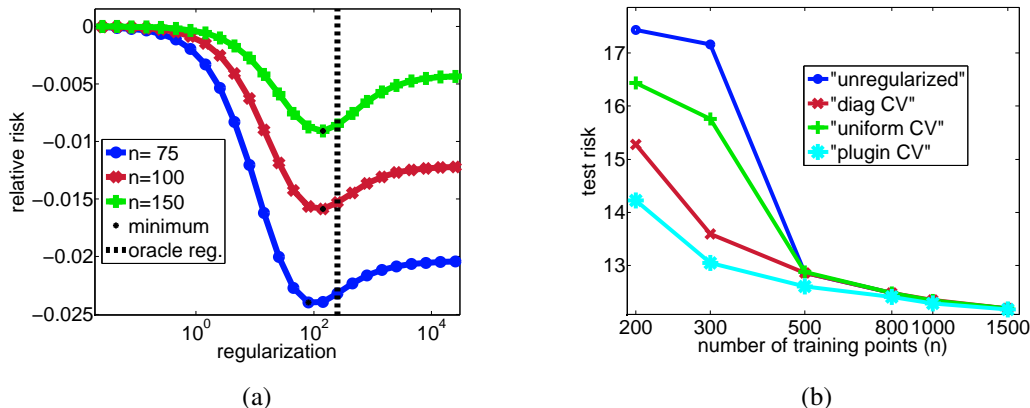

Figure 1: (a) Relative risk ($\mathbb{L}_n(\lambda)$) of the hybrid generative/discriminative estimator for various $\lambda$; the $\lambda$ attaining the minimum of $\mathbb{L}_n(\lambda)$ is close to the oracle $\lambda^*$ (the vertical line). (b) On the MHC-I binding prediction task, test risk for the four multi-task estimators; PLUGINCV (estimating all pairwise task affinities using PLUGIN and cross-validating the strength) works best.

30 independent train/test splits. Multi-task regularization actually performs worse than independent learning (DIAGCV) if we assume all tasks are equally related (UNIFORMCV). By learning the full matrix of task affinities (PLUGINCV), we obtain the best results. Note that setting the $O(K^2)$ entries of $\Lambda$ via cross-validation is not computationally feasible, though other approaches are possible [13].

## 4   Related work and discussion

The subject of choosing regularization parameters has received much attention. Much of the learning theory literature focuses on risk bounds, which approximate the expected risk ($\mathcal{L}(\hat{\theta}_n^\lambda)$) with upper bounds. Our analysis provides a different type of approximation—one that is exact in the first few terms of the expansion. Though we cannot make a precise statement about the risk for any given $n$, exact control over the first few terms offers other advantages, e.g., the ability to compare estimators.

To elaborate further, risk bounds are generally based on the complexity of the hypothesis class, whereas our analysis is based on the variance of the estimator. Vanilla uniform convergence bounds yield worst-case analyses, whereas our asymptotic analysis is tailored to a particular problem ($p^*$ and $\theta_\infty$) and algorithm (estimator). Localization techniques [5], regret analyses [9], and stability-based bounds [8] all allow for some degree of problem- and algorithm-dependence. As bounds, however, they necessarily have some looseness, whereas our analysis provides exact constants, at least the ones associated with the lowest-order terms.

Asymptotics has a rich tradition in statistics. In fact, our methodology of performing a Taylor expansion of the risk is reminiscent of AIC [1]. However, our aim is different: AIC is intended for model selection, whereas we are interested in optimizing regularization parameters. The Stein unbiased risk estimate (SURE) is another method of estimating the expected risk for linear models [21], with generalizations to non-linear models [11].

In practice, cross-validation procedures [10] are quite effective. However, they are only feasible when the number of hyperparameters is very small, whereas our approach can optimize many hyperparameters. Section 3.4 showed that combining the two approaches can be effective.

To conclude, we have developed a general asymptotic framework for analyzing regularization, along with an efficient procedure for choosing regularization parameters. Although we are so far restricted to parametric problems with smooth losses and regularizers, we think that these tools provide a complementary perspective on analyzing learning algorithms to that of risk bounds, deepening our understanding of regularization.

## Footnotes

[1] While we do not explicitly assume convexity of $\ell$ and $R_n$, the local nature of our analysis means that we are essentially working under strong convexity.

[2] This assumption can be weakened. If $\ddot{\mathcal{L}} \not\succ 0$, the parameters can only be estimated up to the row space of $\ddot{\mathcal{L}}$. But since we are interested in the parameters $\theta$ only in terms of $\mathcal{L}(\theta)$, this particular non-identifiability of the parameters is irrelevant.

[3]We performed three-fold cross-validation to select $c$ from 21 candidates in $[10^{-5}, 10^5]$.

# References

[1] H. Akaike. A new look at the statistical model identification. *IEEE Transactions on Automatic Control*, 19:716–723, 1974.

[2] A. Argyriou, T. Evgeniou, and M. Pontil. Multi-task feature learning. In *Advances in Neural Information Processing Systems (NIPS)*, pages 41–48, 2007.

[3] B. Bakker and T. Heskes. Task clustering and gating for Bayesian multitask learning. *Journal of Machine Learning Research*, 4:83–99, 2003.

[4] M. S. Bartlett. Approximate confidence intervals. II. More than one unknown parameter. *Biometrika*, 40:306–317, 1953.

[5] P. L. Bartlett, O. Bousquet, and S. Mendelson. Local Rademacher complexities. *Annals of Statistics*, 33(4):1497–1537, 2005.

[6] G. Bouchard. Bias-variance tradeoff in hybrid generative-discriminative models. In *Sixth International Conference on Machine Learning and Applications (ICMLA)*, pages 124–129, 2007.

[7] G. Bouchard and B. Triggs. The trade-off between generative and discriminative classifiers. In *International Conference on Computational Statistics*, pages 721–728, 2004.

[8] O. Bousquet and A. Elisseeff. Stability and generalization. *Journal of Machine Learning Research*, 2:499–526, 2002.

[9] N. Cesa-Bianchi and G. Lugosi. *Prediction, learning, and games*. Cambridge University Press, 2006.

[10] P. Craven and G. Wahba. Smoothing noisy data with spline functions. estimating the correct degree of smoothing by the method of generalized cross-validation. *Numerische Mathematik*, 31(4):377–403, 1978.

[11] Y. C. Eldar. Generalized SURE for exponential families: Applications to regularization. *IEEE Transactions on Signal Processing*, 57(2):471–481, 2009.

[12] T. Evgeniou, C. Micchelli, and M. Pontil. Learning multiple tasks with kernel methods. *Journal of Machine Learning Research*, 6:615–637, 2005.

[13] L. Jacob, F. Bach, and J. Vert. Clustered multi-task learning: A convex formulation. In *Advances in Neural Information Processing Systems (NIPS)*, pages 745–752, 2009.

[14] W. James and C. Stein. Estimation with quadratic loss. In *Fourth Berkeley Symposium in Mathematics, Statistics, and Probability*, pages 361–380, 1961.

[15] J. A. Lasserre, C. M. Bishop, and T. P. Minka. Principled hybrids of generative and discriminative models. In *Computer Vision and Pattern Recognition (CVPR)*, pages 87–94, 2006.

[16] P. Liang, F. Bach, G. Bouchard, and M. I. Jordan. Asymptotically optimal regularization in smooth parametric models. Technical report, ArXiv, 2010.

[17] P. Liang and M. I. Jordan. An asymptotic analysis of generative, discriminative, and pseudo-likelihood estimators. In *International Conference on Machine Learning (ICML)*, 2008.

[18] A. McCallum, C. Pal, G. Druck, and X. Wang. Multi-conditional learning: Generative/discriminative training for clustering and classification. In *Association for the Advancement of Artificial Intelligence (AAAI)*, 2006.

[19] B. Peters, H. Bui, S. Frankild, M. Nielson, C. Lundegaard, E. Kostem, D. Basch, K. Lamberth, M. Harndahl, W. Fleri, S. S. Wilson, J. Sidney, O. Lund, S. Buus, and A. Sette. A community resource benchmarking predictions of peptide binding to MHC-I molecules. *PLoS Compututational Biology*, 2, 2006.

[20] R. Raina, Y. Shen, A. Ng, and A. McCallum. Classification with hybrid generative/discriminative models. In *Advances in Neural Information Processing Systems (NIPS)*, 2004.

[21] C. M. Stein. Estimation of the mean of a multivariate normal distribution. *Annals of Statistics*, 9(6):1135–1151, 1981.

[22] A. W. van der Vaart. *Asymptotic Statistics*. Cambridge University Press, 1998.

